# From Mixtures of Mixtures to Adaptive Transform Coding

**Cynthia Archer and Todd K. Leen**
Department of Computer Science and Engineering
Oregon Graduate Institute of Science & Technology
20000 N.W. Walker Rd, Beaverton, OR 97006-1000
E-mail: archer, tleen@cse.ogi.edu

## Abstract

We establish a principled framework for adaptive transform coding. Transform coders are often constructed by concatenating an ad hoc choice of transform with suboptimal bit allocation and quantizer design. Instead, we start from a probabilistic latent variable model in the form of a mixture of constrained Gaussian mixtures. From this model we derive a transform coding algorithm, which is a constrained version of the generalized Lloyd algorithm for vector quantizer design. A byproduct of our derivation is the introduction of a new transform basis, which unlike other transforms (PCA, DCT, etc.) is explicitly optimized for coding. Image compression experiments show adaptive transform coders designed with our algorithm improve compressed image signal-to-noise ratio up to 3 dB compared to global transform coding and 0.5 to 2 dB compared to other adaptive transform coders.

## 1 Introduction

Compression algorithms for image and video signals often use *transform coding* as a low-complexity alternative to vector quantization (VQ). Transform coders compress multi-dimensional data by *transforming* the signal vectors to new coordinates and *coding* the transform coefficients independently of one another with scalar quantizers.

The coordinate transform may be fixed a priori as in the discrete cosine transform (DCT). It can also be adapted to the signal statistics using, for example, principal component analysis (PCA), where the goal is to concentrate signal energy in a few signal components. Noting that signals such as images and speech are non-stationary, several researchers have developed non-linear [1, 2] and local linear or adaptive [3, 4] PCA transforms for dimension reduction[1]. None of these transforms are designed to minimize compression distortion nor are they designed in concert with quantizer development.

Several researchers have extended the idea of local linear transforms to transform coding [5, 6, 7]. In these adaptive transform coders, the signal space is partitioned into disjoint regions and a transform and set of scalar quantizers are designed for each region. In our own previous work [7], we use k-means partitioning to define the regions. Dony and Haykin [5] partition the space to minimize dimension-reduction error. Tipping and Bishop [6] use soft partitioning according to a probabilistic rule that reduces, in the appropriate limit, to partitioning by dimension-reduction error. These systems neither design transforms nor partition the signal space with the goal of minimizing compression distortion.

This ad hoc construction contrasts sharply with the solid grounding of vector quantization. Nowlan [8] develops a probabilistic framework for VQ by demonstrating the correspondence between a VQ and a mixture of spherically symmetric Gaussians. In the limit that the mixture component variance goes to zero, the Expectation-Maximization (EM) procedure for fitting the mixture model to data becomes identical to the Linde-Buzo-Gray (LBG) algorithm [9] for vector quantizer design.

This paper develops a similar grounding for both global and adaptive (local) transform coding. We define a constrained mixture of Gaussians model that provides a framework for transform coder design. Our new design algorithm is simply a *constrained* version of the LBG algorithm. It iteratively optimizes the signal space partition, the local transforms, the allocation of coding bits, and the scalar quantizer reproduction values until it reaches a local distortion minimum. This approach leads to two new results, an orthogonal transform and a method of partitioning the signal space, both designed to minimize coding error.

## 2 Global Transform Coder Model

In this section, we develop a *constrained* mixture of Gaussians model that provides a probabilistic framework for *global* transform coding.

### 2.1 Latent Variable Model

A transform coder converts a signal to new coordinates and then codes the coordinate values *independently* of one another with scalar quantizers. To replicate this structure, we envision the data as drawn from a $d$-dimensional latent data space, $S$, in which the density $p(s) = p(s_1, s_2, \ldots, s_d)$ is a product of the marginal densities, $p_J(s_J), \; J = 1 \ldots d$.

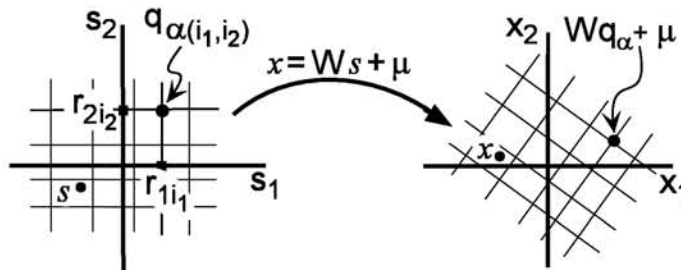

Figure 1: Structure of latent variable space, $S$, and mapping to observed space, $X$. The latent data density consists of a mixture of spherical Gaussians with component means $q_\alpha$ constrained to lie at the vertices of a rectangular grid. The latent data is mapped to the observed space by an orthogonal transform, $W$.

We model the density in the latent space with a constrained mixture of Gaussian densities

$$p(s) = \sum_{\alpha=1}^{K} \pi_\alpha \, p(s|\alpha) \tag{1}$$

where $\pi_\alpha$ are the mixing coefficients and $p(s|\alpha) = \mathcal{N}(q_\alpha, \Sigma_\alpha)$ is Gaussian with mean $q_\alpha$ and variance $\Sigma_\alpha$. The mixture component means, $q_\alpha$, lie at the vertices of a rectangular grid as illustrated in figure (1). The coordinates of $q_\alpha$ are $[r_{1i_1}, r_{2i_2}, \ldots, r_{di_d}]^T$, where $r_{Ji_J}$ is the $i_J^{th}$ grid mark on the $s_J$ axis. There are $K_J$ grid mark values on the $s_J$ axis, so the total number of grid vertices $K = \prod_J K_J$.

We constrain the mixture components variances, $\Sigma_\alpha$ to be spherically symmetric with the same variance, $\sigma^2 \mathbf{I}$, with $\mathbf{I}$ the identity matrix. We do not fit $\sigma^2$ to the data, but treat it as a "knob", which we will turn to zero to reveal a transform coder. These mean and variance constraints yield marginal densities $p_J(s_J|i_J) = \mathcal{N}(r_{Ji_J}, \sigma^2)$. We write the density of $s$ conditioned on $\alpha$ as

$$p(s|\alpha) = p(s_1, \ldots, s_d|\alpha(i_1, \ldots, i_d)) = \prod_{J=1}^{d} p_J(s_J|i_J). \tag{2}$$

and constrain each $\pi_\alpha$ to be a product of prior probabilities, $\pi_{\alpha(i_1,\ldots,i_d)} = \prod_J p_{Ji_J}$.

Incorporating these constraints into (1) and noting that the sum over the mixture components $\alpha$ is equivalent to sums over all grid mark values, the latent density becomes

$$p(s) = \sum_{i_1=1}^{K_1} \sum_{i_2=1}^{K_2} \cdots \sum_{i_d=1}^{K_d} \prod_J p_{Ji_J} \, p_J(s_J|i_J) = \prod_{J=1}^{d} \sum_{i_J=1}^{K_J} p_{Ji_J} \, p_J(s_J|i_J). \tag{3}$$

where the second equality comes by regrouping terms.

The latent data is mapped to the observation space by an orthogonal transformation, $W$ (figure 1). Using $p(x|s) = \delta(x - Ws - \mu)$ and (1), the density on observed data $x$ conditioned on component $\alpha$ is $p(x|\alpha) = \mathcal{N}(Wq_\alpha + \mu, \sigma^2 \mathbf{I})$. The total density on $x$ is

$$p(x) = \sum_{\alpha=1}^{K} \pi_\alpha \, p(x|\alpha) \, . \tag{4}$$

The data log likelihood for $N$ data vectors, $\{x_n, \ n = 1 \ldots N\}$, averaged over the posterior probabilities $p(\alpha|x_n)$ is

$$\mathcal{L} = \sum_{n=1}^{N} \sum_{\alpha=1}^{K} p(\alpha|x_n) \left( \ln \pi_\alpha - \frac{d}{2} \ln(2\pi\sigma^2) - \frac{1}{2\sigma^2} |x_n - Wq_\alpha - \mu|^2 \right) \tag{5}$$

## 2.2 Model Fitting and Transform Coder design

The model (4) can be fit to data using the EM algorithm. In the limit that the variance of the mixture components goes to zero, the EM procedure for fitting the mixture model to data corresponds to a constrained LBG (CLBG) algorithm for optimal transform coder design.

In the limit $\sigma^2 \to 0$ the entropy term, $\ln \pi_\alpha$, becomes insignificant and the component posteriors collapse to

$$p(\alpha|x_n) \to \begin{cases} 1 & \text{if } |x_n - Wq_\alpha - \mu|^2 < |x_n - Wq_\gamma - \mu|^2 \ \forall \gamma \neq \alpha \\ 0 & \text{otherwise} \end{cases} \tag{6}$$

Each data vector is assigned to the component whose mean has the smallest Euclidean distance to it. These assignments minimize mean squared error.

In the limit that $\sigma^2 \to 0$, maximizing the likelihood (5) is equivalent to minimizing compression distortion

$$D = \sum_\alpha \pi_\alpha \frac{1}{N_\alpha} \sum_{x \in R_\alpha} |x - Wq_\alpha - \mu|^2 \tag{7}$$

where $R_\alpha = \{x \,|\, p(\alpha|x) = 1\}$, $N_\alpha$ is the number of $x \in R_\alpha$, and $\pi_\alpha = N_\alpha/N$.

To optimize the transform, we find the orientation of the current quantizer grid which minimizes (7). The transform, $W$, is constrained to be orthogonal, that is $W^T W = \mathbf{I}$. We first define the matrix of outer products $Q$

$$Q = \sum_\alpha \pi_\alpha q_\alpha \left( \frac{1}{N_\alpha} \sum_{x \in R_\alpha} (x - \mu)^T \right) \ . \tag{8}$$

Minimizing the distortion (7) with respect to some element of $W$ and using Lagrange multipliers to enforce the orthogonality of $W$ yields the condition

$$QW = W^T Q^T \tag{9}$$

or $QW$ is symmetric. This symmetry condition and the orthogonality condition, $W^T W = \mathbf{I}$, uniquely determine the coding optimal transform (COT) $W$. The COT reduces to the PCA transform when the data is Gaussian. However, in general the COT differs from PCA. For instance in global transform coding trials on a variety of grayscale images, the COT improves signal-to-noise ratio (SNR) relative to PCA by 0.2 to 0.35 dB for fixed-rate coding at 1.0 bits per pixel (bpp). For variable-rate coding, SNR improvement due to using the COT is substantial, 0.3 to 1.2 dB for entropies of 0.25 to 1.25 bpp.

We next minimize (7) with respect to the grid mark values, $r_{Ji_J}$, for $J = 1 \ldots d$ and $i_J = 1 \ldots K_J$ and the number of grid values $K_J$ for each coordinate. It is advantageous to rewrite compression distortion as the sum of distortions $D = \sum_J D_J$ due to quantizing the transform coefficients $s_J = W_J^T x$, where $W_J$ is the $J^{th}$ column vector of $W$. The $r_{Ji_J}$ grid mark values that minimize each $D_J$ are the reproduction values of a scalar Lloyd quantizer [10] designed for the transform coefficients, $s_J$. $K_J$ is the number of reproduction values in the quantizer for transform coordinate $J$. Allocating the $\log_2(K)$ coding bits among the transform coordinates so that we minimize distortion [11] determines the optimal $K_J$'s.

## 3   Local Transform Coder Model

In this section, we develop a mixture of constrained Gaussian mixtures model that provides a probabilistic framework for *adaptive* transform coding.

### 3.1   Latent Variable Model

A local or adaptive transform coder identifies regions in data space that require different quantizer grids and orthogonal transforms. A separate transform coder is designed for each of these regions. To replicate this structure, we envision the observed data as drawn from one of $M$ grids in the latent space. The latent variables, $s$, are modeled with a mixture of Gaussian densities, where the mixture components are constrained to lie at the grid vertices. Each grid has the same number of mixture components, $K$, however the number and spacing of grid marks on each axis can differ. This is illustrated schematically (in the hard-clustering limit) in figure 2.

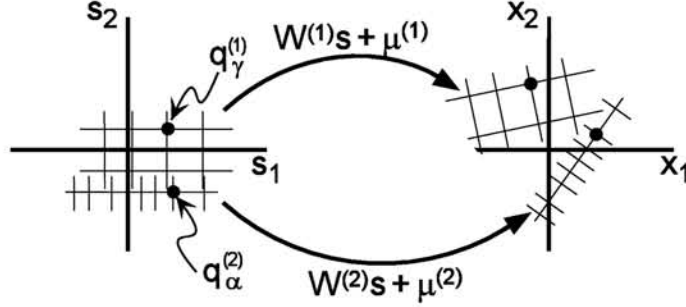

Figure 2: Nonstationary data model : Structure of latent variable space, $S$, and mapping (in the hard clustering limit) to observed space, $X$. The density in the latent space consists of mixtures of spherically symmetric Gaussians. The mixture component means, $q_\alpha^{(m)}$, lie at the vertices the $m^{th}$ grid. Latent data is mapped to the observation space by $W^{(m)}$.

The density on $s$ conditioned on a single mixture component, $\alpha$ in grid m,[2] is $p(s|\alpha, m) = \mathcal{N}(q_\alpha^{(m)}, \sigma^2 \mathbf{I})$. The latent density is a mixture of constrained Gaussian mixture densities,

$$p(s) = \sum_{m=1}^{M} \pi_m \sum_{\alpha=1}^{K} p(\alpha|m) \, p(s|\alpha, m) \qquad (10)$$

The latent data is mapped to the observation space by orthonormal transforms $W^{(m)}$. The density on $x$ conditioned on $\alpha$ in grid m is $p(x|\alpha, m) = \mathcal{N}(W^{(m)} q_\alpha^{(m)} + \mu^{(m)}, \sigma^2 \mathbf{I})$. The observed data density is

$$p(x) = \sum_{m=1}^{M} \pi_m \sum_{\alpha=1}^{K} p(\alpha|m) \, p(x|\alpha, m) \qquad (11)$$

## 3.2  Optimal Adaptive Transform Coder Design

In the limit that $\sigma^2 \to 0$, the EM procedure for fitting this model corresponds to a constrained LBG algorithm for *adaptive* transform coder design. As before, a single mixture component becomes responsible for $x_n$

$$p(\alpha, m|x) \to \begin{cases} 1 & \text{if } |x - W^{(m)} q_\alpha^{(m)} - \mu^{(m)}|^2 \leq |x - W^{(\hat{m})} q_\gamma^{(\hat{m})} - \mu^{(\hat{m})}|^2 \;\; \forall \hat{m}, \gamma \\ 0 & \text{otherwise} \end{cases}$$

$$\tag{12}$$

The coding optimal partition assigns each data vector to the region, m, whose transform coder compresses it with the least distortion. This differs from prior methods that use other partitioning criteria such as K-means clustering or Local PCA partitioning. In K-means clustering, a data vector is assigned to the coder whose mean has the smallest Euclidean distance to it. Local PCA partitions the data space to minimize dimension reduction error [3], not the coding error. Local PCA requires a priori selection of a target dimension, instead of allowing the dimension to be optimized for the desired level of compression.

To minimize distortion with respect to the transform coders, we can optimize the parameters of each region separately. A region's parameters are estimated from just the data vectors assigned to it. We find each region's transform and the number and placement of grid mark values as we did for the global transform coder.

# 4 Adaptive Transform Coding Results

We find the adaptive transform coder for a set of images by applying our algorithm to a training image. The data vectors are $8 \times 8$ image pixel blocks. Then we compress a test image using the resulting transform coder. We measure compressed *test* image quality with signal-to-noise ratio, $\text{SNR} = 10 \log_{10}(\text{pixel variance/MSE})$, where MSE is the per pixel mean-squared coding error.

Our implementation modifies codebook optimization to reduce computational requirements. First, instead of using optimal bit allocation, we use a greedy algorithm [12], which allocates bits one at a time to the coordinate with the largest distortion. In global transform coding trials (0.375 to 0.75 bpp), this substitution reduced SNR by < 0.1 dB. Second, instead of using the coding optimal transform (9), we use the PCA transform. In global transform coding trials (0.25 to 0.75 bpp), this substitution reduced SNR by 0.05 to 0.27 dB.

We report on compression experiments using two types of images, Magnetic Resonance Images (MRI) and gray-scale natural images of traffic moving through street intersections. These MRI images were used by Dony and Haykin in [5] and we duplicate their image pre-processing. One MRI image is decomposed into overlapping $8 \times 8$ blocks to form 15,625 training vectors; a second image is used for testing. The traffic images are frames from two video sequences. We use frames from the first half of both sequences for training and frames from the last halves for testing.

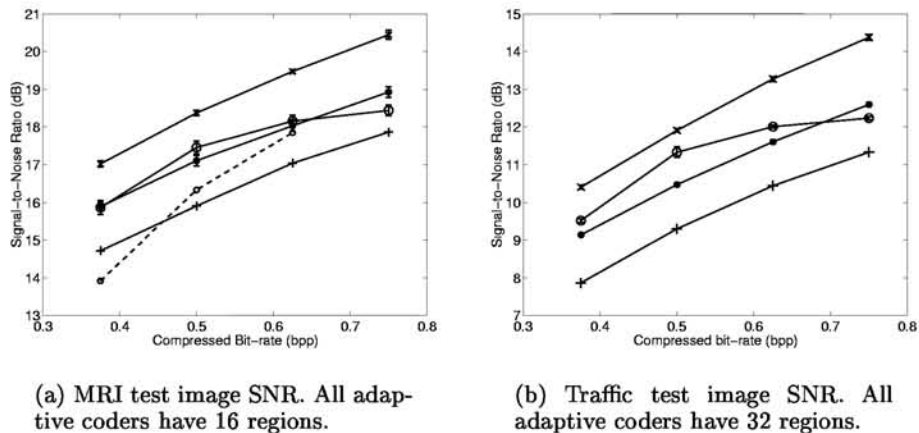

(a) MRI test image SNR. All adaptive coders have 16 regions.

(b) Traffic test image SNR. All adaptive coders have 32 regions.

Figure 3: The × is our coding optimal partition, ○ local PCA partition with dimension eight, • k-means clustering, and + is global PCA. The dotted line values are local PCA results from [5]. Errorbars indicate standard deviation of 8 trials.

Figure 3 shows compressed test image SNR for four compressed bit-rates and four compression methods. The quoted bit-rates *include* the bits necessary to specify region assignments. The × results are for our transform coder which uses coding optimal partitioning. Our system increases SNR compared to global PCA (+) by 2.3 to 3.0 dB, k-means clustering (•) by 1.1 to 1.8 dB and local PCA partitioning with target dimension eight (○) by 0.5 to 2.0 dB. In addition, our system yields image SNRs 1.6 to 3.0 dB higher that Dony and Haykin's local PCA transform coder (dimension eight) [5]. Their local PCA coder does not use optimal bit allocation or Lloyd quantizers, which further reduces compressed image SNR.

# 5 Summary

In this paper, we cast the design of both conventional and adaptive transform coders as a constrained optimization procedure. We derive our algorithm from the EM procedure for fitting a mixture of mixtures model to data. In contrast to standard transform coder design, all operations: partitioning the signal space (for the adaptive case), transform design, allocation of coding bits, and quantizer design, are coupled together to minimize compression distortion. This approach leads to a new transform basis that is optimized for coding. The coding optimal transform is in general different from PCA. This approach also leads to a method of data space partitioning that is optimized for coding. This method assigns each signal vector to the coder the compresses it with the least distortion. Our empirical results show marked SNR improvement (0.5 to 2 dB) relative to other partitioning methods.

**Acknowledgements**

The authors wish to thank Robert Dony and Simon Haykin for the use of their MRI image data and the Institute für Algorithmen und Kognitive Systems, Universität Karlsruhe for making their traffic images available. This work was funded by NSF under grants ECS-9704094 and ECS-9976452.

## Footnotes

[1] In dimension reduction the original $d$-dimensional signal is projected onto a subspace or submanifold of lower dimension. The retained coordinates are *not* quantized.

[2]Each grid has its own mixture component index, $\alpha_m$. We drop the m subscript from $\alpha$ to simplify notation.

# References

[1] Mark A. Kramer. Nonlinear prinipal component analysis using autoassociative neural networks. *AIChE journal*, 37(2):233–243, February 1991.

[2] David DeMers and Garrison Cottrell. Non-linear dimensionality reduction. In Giles, Hanson, and Cowan, editors, *Advances in Neural Information Processing Systems 5*, San Mateo, CA, 1993. Morgan Kaufmann.

[3] Nanda Kambhatla and Todd K. Leen. Fast non-linear dimension reduction. In Cowan, Tesauro, and Alspector, editors, *Advances in Neural Information Processing Systems 6*, pages 152–159. Morgan Kauffmann, Feb 1994.

[4] G. Hinton, M. Revow, and P. Dayan. Recognizing handwritten digits using mixtures of linear models. In Tesauro, Touretzky, and Leen, editors, *Advances in Neural Information Processing Systems 7*, pages 1015–1022. MIT Press, 1995.

[5] Robert D. Dony and Simon Haykin. Optimally adaptive transform coding. *IEEE Transactions on Image Processing*, 4(10):1358–1370, 1995.

[6] M. Tipping and C. Bishop. Mixture of probabilistic principal component analyzers. *Neural Computation*, 11(2):443–483, 1999.

[7] C. Archer and T.K. Leen. Optimal dimension reduction and transform coding with mixture principal components. In *Proceedings of International Joint Conference on Neural Networks*, July 1999.

[8] Steve Nowlan. *Soft Competitive Adaptation: neural network learning algorithms based on fitting statistical mixtures*. PhD thesis, School of Computer Science, Carnegie Mellon University, 1991.

[9] Y. Linde, A. Buzo, and R.M. Gray. An algorithm for vector quantizer design. *IEEE Transactions on Communications*, 28(1):84–95, January 1980.

[10] S. Lloyd. Least square optimization in PCM. *IEEE Transactions on Information Theory*, 28(2):129–137, 1982.

[11] Eve A. Riskin. Optimal bit allocation via the generalized BFOS algorithm. *IEEE Transactions on Information Theory*, 37(2):400–402, 1991.

[12] A. Gersho and R. Gray. *Vector Quantization and Signal Compression*. Kluwer Academic, 1992.
